# Stable adaptive control with online learning

**Andrew Y. Ng**
Stanford University
Stanford, CA 94305, USA

**H. Jin Kim**
Seoul National University
Seoul, Korea

## Abstract

Learning algorithms have enjoyed numerous successes in robotic control tasks. In problems with time-varying dynamics, online learning methods have also proved to be a powerful tool for automatically tracking and/or adapting to the changing circumstances. However, for safety-critical applications such as airplane flight, the adoption of these algorithms has been significantly hampered by their lack of safety, such as "stability," guarantees. Rather than trying to show difficult, a priori, stability guarantees for specific learning methods, in this paper we propose a method for "monitoring" the controllers suggested by the learning algorithm online, and rejecting controllers leading to instability. We prove that even if an *arbitrary* online learning method is used with our algorithm to control a linear dynamical system, the resulting system is stable.

## 1 Introduction

Online learning algorithms provide a powerful set of tools for automatically fine-tuning a controller to optimize performance while in operation, or for automatically adapting to the changing dynamics of a control problem. [2] Although one can easily imagine many complex learning algorithms (SVMs, gaussian processes, ICA, ...,) being powerfully applied to online learning for control, for these methods to be widely adopted for applications such as airplane flight, it is critical that they come with safety guarantees, specifically *stability* guarantees. In our interactions with industry, we also found stability to be a frequently raised concern for online learning. We believe that the lack of safety guarantees represents a significant barrier to the wider adoption of many powerful learning algorithms for online adaptation and control. It is also typically infeasible to replace formal stability guarantees with only empirical testing: For example, to convincingly demonstrate that we can safely fly a fleet of 100 aircraft for 10000 hours would require $10^6$ hours of flight-tests.

The control literature contains many examples of ingenious stability proofs for various online learning schemes. It is impossible to do this literature justice here, but some examples include [10, 7, 12, 8, 11, 5, 4, 9]. However, most of this work addresses only very specific online learning methods, and usually quite simple ones (such as ones that switch between only a finite number of parameter values using a specific, simple, decision rule, e.g., [4]). In this paper, rather than trying to show difficult a priori stability guarantees for specific algorithms, we propose a method for "monitoring" an *arbitrary* learning algorithm being used to control a linear dynamical system. By rejecting control values online that appear to be leading to instability, our algorithm ensures that the resulting controlled system is stable.

## 2 Preliminaries

Following most work in control [6], we will consider control of a linear dynamical system. Let $x_t \in \mathbb{R}^{n_x}$ be the $n_x$ dimensional state at time $t$. The system is initialized to $x_0 = \vec{0}$. At each time $t$, we select a control action $u_t \in \mathbb{R}^{n_u}$, as a result of which the state transitions to

$$x_{t+1} = Ax_t + Bu_t + w_t. \tag{1}$$

Here, $A \in \mathbb{R}^{n_x \times n_x}$ and $B \in \mathbb{R}^{n_x \times n_u}$ govern the dynamics of the system, and $w_t$ is a disturbance term. We will not make any distributional assumptions about the source of the

disturbances $w_t$ for now (indeed, we will consider a setting where an adversary chooses them from some bounded set). For many applications, the controls are chosen as a linear function of the state:

$$u_t = K_t x_t. \qquad (2)$$

Here, the $K_t \in \mathbb{R}^{n_u \times n_x}$ are the control **gains**. If the goal is to minimize the expected value of a quadratic cost function over the states and actions $J = (1/T) \sum_{t=1}^{T} x_t^T Q x_t + u_t^T R u_t$ and the $w_t$ are gaussian, then we are in the LQR (linear quadratic regulation) control setting. Here, $Q \in \mathbb{R}^{n_x \times n_x}$ and $R \in \mathbb{R}^{n_u \times n_u}$ are positive semi-definite matrices. In the infinite horizon setting, under mild conditions there exists an optimal **steady-state** (or **stationary**) gain matrix $K$, so that setting $K_t = K$ for all $t$ minimizes the expected value of $J$. [1]

We consider a setting in which an online learning algorithm (also called an **adaptive control** algorithm) is used to design a controller. Thus, on each time step $t$, an online algorithm may (based on the observed states and action sequence so far) propose some new gain matrix $K_t$. If we follow the learning algorithm's recommendation, then we will start choosing controls according to $u = K_t x$. More formally, an online learning algorithm is a function $f : \cup_{t=1}^{\infty} (\mathbb{R}^{n_x} \times \mathbb{R}^{n_u})^t \mapsto \mathbb{R}^{n_u \times n_x}$ mapping from finite sequences of states and actions $(x_0, u_0, \ldots, x_{t-1}, u_{t-1})$ to controller gains $K_t$. We assume that $f$'s outputs are bounded ($||K_t||_F \leq \psi$ for some $\psi > 0$, where $|| \cdot ||_F$ is the Frobenius norm).

## 2.1 Stability

In classical control theory [6], probably the most important desideratum of a controlled system is that it must be stable. Given a fixed adaptive control algorithm $f$ and a fixed sequence of disturbance terms $w_0, w_1, \ldots$, the sequence of states $x_t$ visited is exactly determined by the equations

$$K_t = f(x_0, u_0, \ldots, x_{t-1}, u_{t-1}); \quad x_{t+1} = A x_t + B \cdot K_t x_t + w_t. \quad t = 0, 1, 2, \ldots \quad (3)$$

Thus, for fixed $f$, we can think of the (controlled) dynamical system as a mapping from the sequence of disturbance terms $w_t$ to the sequence of states $x_t$. We now give the most commonly-used definition of stability, called BIBO stability (see, e.g., [6]).

**Definition.** A system controlled by $f$ is bounded-input bounded-output (BIBO) stable if, given any constant $c_1 > 0$, there exists some constant $c_2 > 0$ so that for all sequences of disturbance terms satisfying $||w_t||_2 \leq c_1$ (for all $t = 1, 2, \ldots$), the resulting state sequence satisfies $||x_t||_2 \leq c_2$ (for all $t = 1, 2, \ldots$).

Thus, a system is BIBO stable if, under bounded disturbances to it (possibly chosen by an adversary), the state remains bounded and does not diverge.

We also define the $t$-th step **dynamics matrix** $D_t$ to be $D_t = A + B K_t$. Note therefore that the state transition dynamics of the system (right half of Equation 3) may now be written $x_{t+1} = D_t x_t + w_t$. Further, the dependence of $x_t$ on the $w_t$'s can be expressed as follows:

$$\begin{aligned} x_t &= w_{t-1} + D_{t-1} x_{t-1} = w_{t-1} + D_{t-1}(w_{t-2} + D_{t-2} x_{t-2}) = \cdots && (4) \\ &= w_{t-1} + D_{t-1} w_{t-2} + D_{t-1} D_{t-2} w_{t-3} + \cdots + D_{t-1} \cdots D_1 w_0. && (5) \end{aligned}$$

Since the number of terms in the sum above grows linearly with $t$, to ensure BIBO stability of a system—i.e., that $x_t$ remains bounded for all $t$—it is usually necessary for the terms in the sum to decay rapidly, so that the sum remains bounded. For example, if it were true that $||D_{t-1} \cdots D_{t-k+1} w_{t-k}||_2 \leq (1 - \epsilon)^k$ for some $0 < \epsilon < 1$, then the terms in the sequence above would be norm bounded by a geometric series, and thus the sum is bounded. More generally, the disturbance $w_t$ contributes a term $D_{t+k-1} \cdots D_{t+1} w_t$ to the state $x_{t+k}$, and we would like $D_{t+k-1} \cdots D_{t+1} w_t$ to become small rapidly as $k$ becomes large (or, in the control parlance, for the effects of the disturbance $w_t$ on $x_{t+k}$ to be **attenuated** quickly).

If $K_t = K$ for all $t$, then we say that we using a (nonadaptive) stationary controller $K$. In this setting, it is straightforward to check if our system is stable. Specifically, it is BIBO stable if and only if the magnitude of all the eigenvalues of $D = D_t = A + B K_t$ are strictly less than 1. [6] To informally see why, note that the effect of $w_t$ on $x_{t+k}$ can be written $D^{k-1} w_t$ (as in Equation 5). Moreover, $|\lambda_{\max}(D)| < 1$ implies $D^{k-1} w_t \to 0$ as $k \to \infty$. Thus, the disturbance $w_t$ has a negligible influence on $x_{t+k}$ for large $k$. More precisely, it

is possible to show that, under the assumption that $||w_t|| \leq c_1$, the sequence on the right hand side of (5) is upper-bounded by a geometrically decreasing sequence, and thus its sum must also be bounded. [6]

It was easy to check for stability when $K_t$ was stationary, because the mapping from the $w_t$'s to the $x_t$'s was linear. In more general settings, if $K_t$ depends in some complex way on $x_1, \ldots, x_{t-1}$ (which in turn depend on $w_0, \ldots, w_{t-2}$), then $x_{t+1} = Ax_t + BK_t x_t + w_t$ will be a nonlinear function of the sequence of disturbances.[1] This makes it significantly more difficult to check for BIBO stability of the system.

Further, unlike the stationary case, it is well-known that $\lambda_{\max}(D_t) < 1$ (for all $t$) is insufficient to ensure stability. For example, consider a system where $D_t = D_{\mathrm{odd}}$ if $t$ is odd, and $D_t = D_{\mathrm{even}}$ otherwise, where[2]

$$D_{\mathrm{odd}} = \begin{bmatrix} 0.9 & 0 \\ 10 & 0.9 \end{bmatrix}; \quad D_{\mathrm{even}} = \begin{bmatrix} 0.9 & 10 \\ 0 & 0.9 \end{bmatrix}. \tag{6}$$

Note that $\lambda_{\max}(D_t) = 0.9 < 1$ for all $t$. However, if we pick $w_0 = [1\ 0]^T$ and $w_1 = w_2 = \ldots = 0$, then (following Equation 5) we have

$$x_{2t+1} = D_{2t} D_{2t-1} D_{2t-2} \ldots D_2 D_1 w_0 \tag{7}$$

$$= (D_{\mathrm{even}} D_{\mathrm{odd}})^t w_0 \tag{8}$$

$$= \begin{bmatrix} 100.81 & 9 \\ 9 & 0.81 \end{bmatrix}^t w_0 \tag{9}$$

Thus, even though the $w_t$'s are bounded, we have $||x_{2t+1}||_2 \geq (100.81)^t$, showing that the state sequence is not bounded. Hence, this system is not BIBO stable.

# 3 Checking for stability

If $f$ is a complex learning algorithm, it is typically very difficult to guarantee that the resulting system is BIBO stable. Indeed, even if $f$ switches between only two specific sets of gains $K$, and if $w_0$ is the only non-zero disturbance term, it can still be undecidable to determine whether the state sequence remains bounded. [3] Rather than try to give a priori guarantees on $f$, we instead propose a method for ensuring BIBO stability of a system by "monitoring" the control gains proposed by $f$, and rejecting gains that appear to be leading to instability. We start computing controls according to a set of gains $\hat{K}_t$ only if it is accepted by the algorithm.

From the discussion in Section 2.1, the criterion for accepting or rejecting a set of gains $\hat{K}_t$ cannot simply be to check if $\lambda_{\max}(A + BK_t) = \lambda_{\max}(D_t) < 1$. Specifically, $\lambda_{\max}(D_2 D_1)$ is not bounded by $\lambda_{\max}(D_2)\lambda_{\max}(D_1)$, and so even if $\lambda_{\max}(D_t)$ is small for all $t$—which would be the case if the gains $K_t$ for any fixed $t$ could be used to obtain a stable stationary controller—the quantity $\lambda_{\max}(\prod_{\tau=1}^{t} D_\tau)$ can still be large, and thus $(\prod_{\tau=1}^{t} D_\tau)w_0$ can be large. However, the following holds for the largest singular value $\sigma_{\max}$ of matrices. Though the result is quite standard, for the sake of completeness we include a proof.[3]

**Proposition 3.1:** *Let any matrices $P \in \mathbb{R}^{l \times m}$ and $Q \in \mathbb{R}^{m \times n}$ be given. Then $\sigma_{\max}(PQ) \leq \sigma_{\max}(P)\sigma_{\max}(Q)$.*

**Proof.** $\sigma_{\max}(PQ) = \max_{u,v:||u||_2=||v||_2=1} u^T PQv$. Let $u^*$ and $v^*$ be a pair of vectors attaining the maximum in the previous equation. Then $\sigma_{\max}(PQ) = u^{*T} PQ v^* \leq ||u^{*T}P||_2 \cdot ||Qv^*||_2 \leq \max_{v,u:||v||_2=||u||_2=1} ||u^T P||_2 \cdot ||Qv||_2 = \sigma_{\max}(P)\sigma_{\max}(Q)$. □

Thus, if we could ensure that $\sigma_{\max}(D_t) \leq 1 - \epsilon$ for all $t$, we would find that the influence of $w_0$ on $x_t$ has norm bounded by $||D_{t-1} D_{t-2} \ldots D_1 w_0||_2 = \sigma_{\max}(D_{t-1} \ldots D_1 w_0) \leq$

$\sigma_{\max}(D_{t-1})\ldots\sigma_{\max}(D_1)\|w_0\|_2 \leq (1-\epsilon)^{t-1}\|w_0\|_2$ (since $\|v\|_2 = \sigma_{\max}(v)$ if $v$ is a vector). Thus, the influence of $w_t$ on $x_{t+k}$ goes to 0 as $k \to \infty$.

However, it would be an overly strong condition to demand that $\sigma_{\max}(D_t) < 1-\epsilon$ for every $t$. Specifically, there are many stable, stationary controllers that do not satisfy this. For example, either one of the matrices $D_t$ in (6), if used as the stationary dynamics, is stable (since $\lambda_{\max} = 0.9 < 1$). Thus, it should be acceptable for us to use a controller with either of these $D_t$ (so long as we do not switch between them on every step). But, these $D_t$ have $\sigma_{\max} \approx 10.1 > 1$, and thus would be rejected if we were to demand that $\sigma_{\max}(D_t) < 1-\epsilon$ for every $t$. Thus, we will instead ask only for a weaker condition, that for all $t$,
$$\sigma_{\max}(D_t \cdot D_{t-1} \cdots D_{t-N+1}) < 1 - \epsilon. \tag{10}$$
This is motivated by the following, which shows that any stable, stationary controller meets this condition (for sufficiently large $N$):

**Proposition 3.2:** *Let any $0 < \epsilon < 1$ and any $D$ with $\lambda_{\max}(D) < 1$ be given. Then there exists $N_0 > 0$ so that for all $N \geq N_0$, we have that $\sigma_{\max}(D^N) \leq 1 - \epsilon$.*

The proof follows from the fact that $\lambda_{\max}(D) < 1$ implies $D^N \to 0$ as $N \to \infty$. Thus, given any fixed, stable controller, if $N$ is sufficiently large, it will satisfy (10). Further, if (10) holds, then $w_0$'s influence on $x_{kN+1}$ is bounded by
$$
\begin{aligned}
\|D_{kN} \cdot D_{kN-1} \cdots D_1 w_0\|_2 &\leq \sigma_{\max}(D_{kN} \cdot D_{kN-1} \cdots D_1)\|w_0\|_2 \\
&\leq \prod_{i=0}^{k-1} \sigma_{\max}(D_{iN+N} D_{iN+N-1} \cdots D_{iN+1})\|w_0\|_2 \\
&\leq (1-\epsilon)^k \|w_0\|_2, \tag{11}
\end{aligned}
$$
which goes to 0 geometrically quickly as $k \to \infty$. (The first and second inequalities above follow from Proposition 3.1.) Hence, the disturbances' effects are attenuated quickly.

To ensure that (10) holds, we propose the following algorithm. Below, $N > 0$ and $0 < \epsilon < 1$ are parameters of the algorithm.

1. Initialization: Assume we have some initial stable controller $K_0$, so that $\lambda_{\max}(D_0) < 1$, where $D_0 = A + BK_0$. Also assume that $\sigma_{\max}(D_0^N) \leq 1 - \epsilon$.[4] Finally, for all values of $\tau < 0$. define $K_\tau = K_0$ and $D_\tau = D_0$.
2. For $t = 1, 2, \ldots$
   (a) Run the online learning algorithm $f$ to compute the next set of proposed gains $\hat{K}_t = f(x_0, u_0, \ldots, x_{t-1}, u_{t-1})$.
   (b) Let $\hat{D}_t = A + B\hat{K}_t$, and check if
   $$
   \begin{aligned}
   \sigma_{\max}(\hat{D}_t D_{t-1} D_{t-2} D_{t-3} \ldots D_{t-N+1}) &\leq 1-\epsilon \tag{12} \\
   \sigma_{\max}(\hat{D}_t^2 D_{t-1} D_{t-2} \ldots D_{t-N+2}) &\leq 1-\epsilon \tag{13} \\
   \sigma_{\max}(\hat{D}_t^3 D_{t-1} \ldots D_{t-N+3}) &\leq 1-\epsilon \tag{14} \\
   &\cdots \\
   \sigma_{\max}(\hat{D}_t^N) &\leq 1-\epsilon \tag{15}
   \end{aligned}
   $$
   (c) If all of the $\sigma_{\max}$'s above are less than $1 - \epsilon$, we ACCEPT $\hat{K}_t$, and set $K_t = \hat{K}_t$. Otherwise, REJECT $\hat{K}_t$, and set $K_t = K_{t-1}$.
   (d) Let $D_t = A + BK_t$, and pick our action at time $t$ to be $u_t = K_t x_t$.

We begin by showing that, if we use this algorithm to "filter" the gains output by the online learning algorithm, Equation (10) holds.

**Lemma 3.3:** *Let $f$ and $w_0, w_1, \ldots$ be arbitrary, and let $K_0, K_1, K_2, \ldots$ be the sequence of gains selected using the algorithm above. Let $D_t = A + BK_t$ be the corresponding dynamics matrices. Then for every $-\infty < t < \infty$, we have*[5]
$$\sigma_{\max}(D_t \cdot D_{t-1} \cdots D_{t-N+1}) \leq 1 - \epsilon. \tag{16}$$

**Proof.** Let any $t$ be fixed, and let $\tau = \max(\{0\} \cup \{t' : 1 \le t' \le t, \hat{K}_{t'}\text{was accepted}\})$. Thus, $\tau$ is the index of the time step at which we most recently accepted a set of gains from $f$ (or 0 if no such gains exist). So, $K_\tau = K_{\tau+1} = \ldots = K_t$, since the gains stay the same in every time step on which we do not accept a new one. This also implies

$$D_\tau = D_{\tau+1} = \ldots = D_t. \tag{17}$$

We will treat the cases (i) $\tau = 0$, (ii) $1 \le \tau \le t - N + 1$ and (iii) $\tau > t - N + 1$, $\tau \ge 1$ separately. In case (i), $\tau = 0$, and we did not accept any gains after time 0. Thus $K_t = \cdots = K_{t-N+1} = K_0$, which implies $D_t = \cdots = D_{t-N+1} = D_0$. But from Step 1 of the algorithm, we had chosen $N$ sufficiently large that $\sigma_{\max}(D_0^N) \le 1 - \epsilon$. This shows (16). In case (ii), $\tau \le t - N + 1$ (and $\tau > 0$). Together with (17), this implies

$$D_t \cdot D_{t-1} \cdot \cdots \cdot D_{t-N+1} = D_\tau^N. \tag{18}$$

But $\sigma_{\max}(D_\tau^N) \le 1 - \epsilon$, because at time $\tau$, when we accepted $K_\tau$, we would have checked that Equation (15) holds. In case (iii), $\tau > t - N + 1$ (and $\tau > 0$). From (17) we have

$$D_t \cdot D_{t-1} \cdot \cdots \cdot D_{t-N+1} = D_\tau^{t-\tau+1} \cdot D_{\tau-1} \cdot D_{\tau-2} \cdot \cdots \cdot D_{t-N+1}. \tag{19}$$

But when we accepted $K_\tau$, we would have checked that (12-15) hold, and the $t - \tau + 1$-st equation in (12-15) is exactly that the largest singular value of (19) is at most $1 - \epsilon$. $\square$

**Theorem 3.4:** *Let an arbitrary learning algorithm $f$ be given, and suppose we use $f$ to control a system, but using our algorithm to accept/reject gains selected by $f$. Then, the resulting system is BIBO stable.*

**Proof.** Suppose $||w_t||_2 \le c_1$ for all $t$. For convenience also define $w_{-1} = w_{-2} = \cdots = 0$, and let $\psi' = ||A||_F + \psi||B||_F$. From (5),

$$
\begin{aligned}
||x_t||_2 &= ||\textstyle\sum_{k=0}^{\infty} D_{t-1}D_{t-2}\cdots D_{t-k}w_{t-k-1}||_2 \\
&\le c_1 \textstyle\sum_{k=0}^{\infty} ||D_{t-1}D_{t-2}\cdots D_{t-k}||_2 \\
&= c_1 \textstyle\sum_{j=0}^{\infty}\sum_{k=0}^{N-1} \sigma_{\max}(D_{t-1}D_{t-2}\cdots D_{t-jN-k}) \\
&\le c_1 \textstyle\sum_{j=0}^{\infty}\sum_{k=0}^{N-1} \sigma_{\max}((\prod_{l=0}^{j-1} D_{t-lN-1}D_{t-lN-2}\cdots D_{t-lN-N}) \\
&\qquad \cdot D_{t-jN-1}\cdots D_{t-jN-k}) \\
&\le c_1 \textstyle\sum_{j=0}^{\infty}\sum_{k=0}^{N-1} (1-\epsilon)^j \cdot \sigma_{\max}(D_{t-jN-1}\cdots D_{t-jN-k}) \\
&\le c_1 \textstyle\sum_{j=0}^{\infty}\sum_{k=0}^{N-1} (1-\epsilon)^j \cdot (\psi')^k \\
&\le c_1 \tfrac{1}{\epsilon} N(1+\psi')^N
\end{aligned}
$$

The third inequality follows from Lemma 3.3, and the fourth inequality follows from our assumption that $||K_t||_F \le \psi$, so that $\sigma_{\max}(D_t) \le ||D_t||_F \le ||A||_F + ||B||_F||K_t||_F \le ||A||_F + \psi||B_t||_F = \psi'$. Hence, $||x_t||_2$ remains uniformly bounded for all $t$. $\square$

Theorem 3.4 guarantees that, using our algorithm, we can safely apply *any* adaptive control algorithm $f$ to our system. As discussed previously, it is difficult to exactly characterize the class of BIBO-stable controllers, and thus the set of controllers that we can safety accept. However, it is possible to show a partial converse to Theorem 3.4 that certain large, "reasonable" classes of adaptive control methods will always have their proposed controllers accepted by our method. For example, it is a folk theorem in control that if we use only stable sets of gains ($K : \lambda_{\max}(A + BK) < 1$), and if we switch "sufficiently slowly" between them, then system will be stable. For our specific algorithm, we can show the following:

**Theorem 3.5:** *Let any $0 < \epsilon < 1$ be fixed, and let $\mathcal{K} \subseteq \mathbb{R}^{n_u \times n_x}$ be a finite set of controller gains, so that for all $K \in \mathcal{K}$, we have $\lambda_{\max}(A + BK) < 1$. Then there exist constants $N_0$ and $k$ so that for all $N \ge N_0$, if (i) Our algorithm is run with parameters $N, \epsilon$, and (ii) The adaptive control algorithm $f$ picks only gains in $\mathcal{K}$, and moreover switches gains no more than once every $k$ steps (i.e., $\hat{K}_t \ne \hat{K}_{t+1} \Rightarrow \hat{K}_{t+1} = \hat{K}_{t+2} = \cdots = \hat{K}_{t+k}$), then all controllers proposed by $f$ will be accepted.*

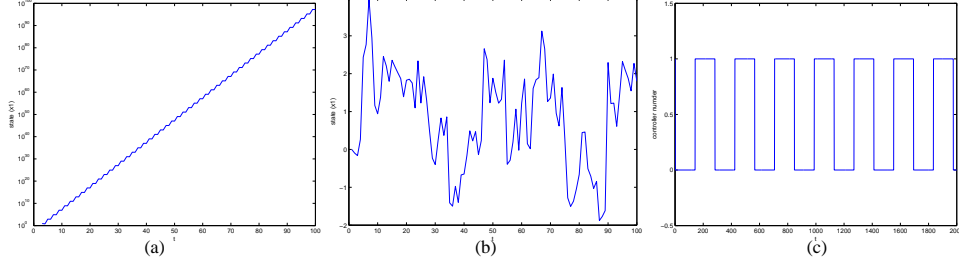

Figure 1: (a) Typical state sequence (first component $x_{t,1}$ of state vector) using switching controllers from Equation (6). (Note log-scale on vertical axis.) (b) Typical state sequence using our algorithm and the same controller $f$. ($N = 150, \epsilon = 0.1$) (c) Index of the controller used over time, when using our algorithm.

The proof is omitted due to space constraints. A similar result also holds if $\mathcal{K}$ is infinite (but $\exists c > 0, \ \forall K \in \mathcal{K}, \ \lambda_{\max}(A + BK) \leq 1 - c$), and if the proposed gains change on every step but the differences $||\hat{K}_t - \hat{K}_{t+1}||_F$ between successive values is small.

## 4 Experiments

We now present experimental results illustrating the behavior of our algorithm. In the first experiment, we apply the switching controller given in (6). Figure 1a shows a typical state sequence resulting from using this controller without using our algorithm to monitor it (and $w_t$'s from an IID standard Normal distribution). Even though $\lambda_{\max}(D_t) < 1$ for all $t$, the controlled system is unstable, and the state rapidly diverges. In contrast, Figure 1b shows the result of rerunning the same experiment, but using our algorithm to accept or reject controllers. The resulting system is stable, and the states remain small. Figure 1c also shows which of the two controllers in (6) is being used at each time, when our algorithm is used. (If do not use our algorithm so that the controller switches on every time step, this figure would switch between 0 and 1 on every time step.) We see that our algorithm is rejecting most of the proposed switches to the controller; specifically, it is permitting $f$ to switch between the two controllers only every 140 steps or so. By slowing down the rate at which we switch controllers, it causes the system to become stable (compare Theorem 3.5).

In our second example, we will consider a significantly more complex setting representative of a real-world application. We consider controlling a Boeing 747 aircraft in a setting where the states are only partially observable. We have a four-dimensional state vector $x_t$ consisting of the sideslip angle $\beta$, bank angle $\phi$, yaw rate, and roll rate of the aircraft in cruise flight. The two-dimensional controls $u_t$ are the rudder and aileron deflections. The state transition dynamics are given as in Equation (1)[6] with IID gaussian disturbance terms $w_t$. But instead of observing the states directly, on each time step $t$ we observe only

$$y_t = Cx_t + v_t, \tag{20}$$

where $y_t \in \mathbb{R}^{n_y}$, and the disturbances $v_t \in \mathbb{R}^{n_y}$ are distributed $\text{Normal}(\vec{0}, \Sigma_v)$. If the system is stationary (i.e., if $A, B, C, \Sigma_v, \Sigma_w$ were fixed), then this is a standard LQG problem, and optimal estimates $\hat{x}_t$ of the hidden states $x_t$ are obtained using a Kalman filter:

$$\hat{x}_{t+1} = L_t(y_{t+1} - C(Ax_t + Bu_t)) + A\hat{x}_t + Bu_t, \tag{21}$$

where $L_t \in \mathbb{R}^{n_x \times n_y}$ is the Kalman filter gain matrix. Further, it is known that, in LQG, the optimal steady state controller is obtained by picking actions according to $u_t = K_t \hat{x}_t$, where $K_t$ are appropriate control gains. Standard algorithms exist for solving for the optimal steady-state gain matrices $L$ and $K$. [1]

In our aircraft control problem, $C = \left[ \begin{smallmatrix} 0 & 1 & 0 & 0 \\ 0 & 0 & 0 & 1 \end{smallmatrix} \right]$, so that only two of the four state variables and are observed directly. Further, the noise in the observations varies over time. Specifically, sometimes the variance of the first observation is $\Sigma_{v,11} = \text{Var}(v_{t,1}) = 2$

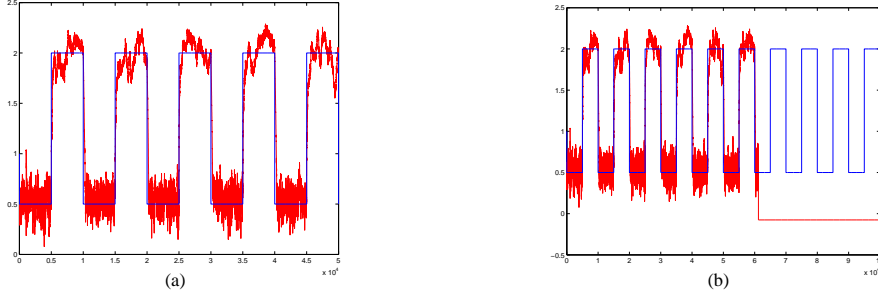

<div align="center">(a)                              (b)</div>

Figure 2: (a) Typical evolution of true $\Sigma_{v,11}$ over time (straight lines) and online approximation to it. (b) Same as (a), but showing an example in which the learned variance estimate became negative.

while the variance of the second observation is $\Sigma_{v,22} = \text{Var}(v_{t,2}) = 0.5$; and sometimes the values of the variances are reversed $\Sigma_{v,11} = 0.5$, $\Sigma_{v,22} = 2$. ($\Sigma_v \in \mathbb{R}^{2\times2}$ is diagonal in all cases.) This models a setting in which, at various times, either of the two sensors may be the more reliable/accurate one.

Since the reliability of the sensors changes over time, one might want to apply an online learning algorithm (such as online stochastic gradient ascent) to dynamically estimate the values of $\Sigma_{v,11}$ and $\Sigma_{v,22}$. Figure 2 shows a typical evolution of $\Sigma_{v,11}$ over time, and the result of using a stochastic gradient ascent learning algorithm to estimate $\Sigma_{v,11}$. Empirically, a stochastic gradient algorithm seems to do fairly well at tracking the true $\Sigma_{v,11}$. Thus, one simple adaptive control scheme would be to take the current estimate of $\Sigma_v$ at each time step $t$, apply a standard LQG solver giving this estimate (and $A, B, C, \Sigma_w$) to it to obtain the optimal steady-state Kalman filter and control gains, and use the values obtained as our proposed gains $L_t$ and $K_t$ for time $t$. This gives a simple method for adapting our controller and Kalman filter parameters to the varying noise parameters.

The adaptive control algorithm that we have described is sufficiently complex that it is extremely difficult to prove that it gives a stable controller. Thus, to guarantee BIBO stability of the system, one might choose to run it with our algorithm. To do so, note that the "state" of the controlled system at each time step is fully characterized by the true world state $x_t$ and the internal state estimate of the Kalman filter $\hat{x}_t$. So, we can define an augmented state vector $\tilde{x}_t = [x_t; \hat{x}_t] \in \mathbb{R}^8$. Because $x_{t+1}$ is linear in $u_t$ (which is in turn linear in $\hat{x}_t$) and similarly $\hat{x}_{t+1}$ is linear in $x_t$ and $u_t$ (substitute (20) into (21)), for a fixed set of gains $K_t$ and $L_t$, we can express $\tilde{x}_{t+1}$ as a linear function of $\tilde{x}_t$ plus a disturbance:

$$\tilde{x}_{t+1} = \tilde{D}_t \tilde{x}_t + \tilde{w}_t. \tag{22}$$

Here, $\tilde{D}_t$ depends implicitly on $A, B, C, L_t$ and $K_t$. (The details are not complex, but are omitted due to space). Thus, if a learning algorithm is proposing new $\hat{K}_t$ and $\hat{L}_t$ matrices on each time step, we can ensure that the resulting system is BIBO stable by computing the corresponding $\tilde{D}_t$ as a function of $\hat{K}_t$ and $\hat{L}_t$, and running our algorithm (with $\tilde{D}_t$'s replacing the $D_t$'s) to decide if the proposed gains should be accepted. In the event that they are rejected, we set $K_t = K_{t-1}$, $L_t = L_{t-1}$.

It turns out that there is a very subtle bug in the online learning algorithm. Specifically, we were using standard stochastic gradient ascent to estimate $\Sigma_{v,11}$ (and $\Sigma_{v,22}$), and on every step there is a small chance that the gradient update overshoots zero, causing $\Sigma_{v,11}$ to become negative. While the probability of this occurring on any particular time step is small, a Boeing 747 flown for sufficiently many hours using this algorithm will eventually encounter this bug and obtain an invalid, negative, variance estimate. When this occurs, the Matlab LQG solver for the steady-state gains outputs $L = 0$ on this and all successive time steps.[7] If this were implemented on a real 747, this would cause it to ignore all observations (Equation 21), enter divergent oscillations (see Figure 3a), and crash. However, using our algorithm, the behavior of the system is shown in Figure 3b. When the learning algorithm

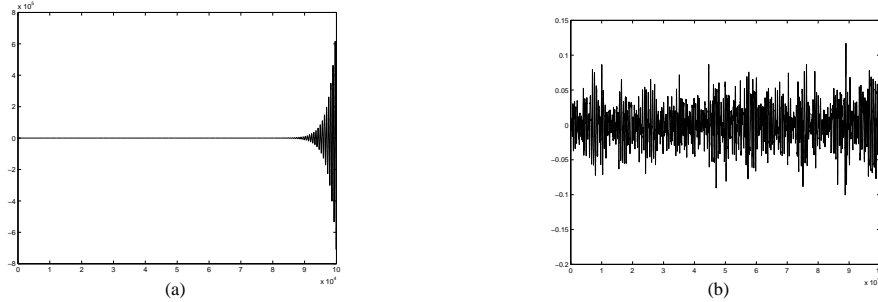

(a)                                                     (b)

Figure 3: (a) Typical plot of state $(x_{t,1})$ using the (buggy) online learning algorithm in a sequence in which $L$ was set to zero part-way through the sequence. (Note scale on vertical axis; this plot is typical of a linear system entering divergent/unstable oscillations.) (b) Results on same sequence of disturbances as in (a), but using our algorithm.

encounters the bug, our algorithm successfully rejects the changes to the gains that lead to instability, thereby keeping the system stable.

## 5  Discussion

Space constraints preclude a full discussion, but these ideas can also be applied to verifying the stability of certain nonlinear dynamical systems. For example, if the $A$ (and/or $B$) matrix depends on the current state but is always expressible as a convex combination of some fixed $A_1, \ldots, A_k$, then we can guarantee BIBO stability by ensuring that (10) holds for all combinations of $D_t = A_i + BK_t$ defined using any $A_i$ $(i = 1, \ldots k)$.[8] The same idea also applies to settings where $A$ may be changing (perhaps adversarially) within some bounded set, or if the dynamics are unknown so that we need to verify stability with respect to a *set* of possible dynamics. In simulation experiments of the Stanford autonomous helicopter, by using a linearization of the non-linear dynamics, our algorithm was also empirically successful at stabilizing an adaptive control algorithm that normally drives the helicopter into unstable oscillations.

## Footnotes

[1]Even if $f$ is linear in its inputs so that $K_t$ is linear in $x_1, \ldots, x_{t-1}$, the state sequence's dependence on $(w_0, w_1, \ldots)$ is still nonlinear because of the multiplicative term $K_t x_t$ in the dynamics (Equation 3).

[2]Clearly, such as system can be constructed with appropriate choices of $A$, $B$ and $K_t$.

[3]The largest singular value of $M$ is $\sigma_{\max}(M) = \sigma_{\max}(M^T) = \max_{u,v:||u||_2=||v||_2=1} u^T Mv = \max_{u:||u||_2=1} ||Mu||_2$. If $x$ is a vector, then $\sigma_{\max}(x)$ is just the $L_2$-norm of $x$.

[4]From Proposition 3.2, it must be possible to choose $N$ satisfying this.

[5]As in the algorithm description, $D_t = D_0$ for $t < 0$.

[6]The parameters $A \in \mathbb{R}^{4 \times 4}$ and $B \in \mathbb{R}^{4 \times 2}$ are obtained from a standard 747 ('yaw damper') model, which may be found in, e.g., the Matlab control toolbox, and various texts such as [6].

[7]Even if we had anticipated this specific bug and clipped $\Sigma_{v,11}$ to be non-negative, the LQG solver (from the Matlab controls toolbox) still outputs invalid gains, since it expects nonsingular $\Sigma_v$.

[8]Checking all $k^N$ such combinations takes time exponential in $N$, but it is often possible to use very small values of $N$, sometimes including $N = 1$, if the states $x_t$ are linearly reparameterized $(x'_t = Mx_t)$ to minimize $\sigma_{\max}(D_0)$.

## References

[1] B. Anderson and J. Moore. *Optimal Control: Linear Quadratic Methods*. Prentice-Hall, 1989.

[2] Karl Astrom and Bjorn Wittenmark. *Adaptive Control (2nd Edition)*. Addison-Wesley, 1994.

[3] V. D. Blondel and J. N. Tsitsiklis. The boundedness of all products of a pair of matrices is undecidable. *Systems and Control Letters*, 41(2):135–140, 2000.

[4] Michael S. Branicky. Analyzing continuous switching systems: Theory and examples. In *Proc. American Control Conference*, 1994.

[5] Michael S. Branicky. Stability of switched and hybrid systems. In *Proc. 33rd IEEE Conf. Decision Control*, 1994.

[6] G. Franklin, J. Powell, and A. Emani-Naeini. *Feedback Control of Dynamic Systems*. Addison-Wesley, 1995.

[7] M. Johansson and A. Rantzer. On the computation of piecewise quadratic lyapunov functions. In *Proceedings of the 36th IEEE Conference on Decision and Control*, 1997.

[8] H. Khalil. *Nonlinear Systems (3rd ed)*. Prentice Hall, 2001.

[9] Daniel Liberzon, João Hespanha, and A. S. Morse. Stability of switched linear systems: A lie-algebraic condition. *Syst. & Contr. Lett.*, 3(37):117–122, 1999.

[10] J. Nakanishi, J.A. Farrell, and S. Schaal. A locally weighted learning composite adaptive controller with structure adaptation. In *International Conference on Intelligent Robots*, 2002.

[11] T. J. Perkins and A. G. Barto. Lyapunov design for safe reinforcement learning control. In *Safe Learning Agents: Papers from the 2002 AAAI Symposium*, pages 23–30, 2002.

[12] Jean-Jacques Slotine and Weiping Li. *Applied Nonlinear Control*. Prentice Hall, 1990.

